# A Bayesian Analysis of Dynamics in Free Recall

**Richard Socher**
Department of Computer Science
Stanford University
Stanford, CA 94305
richard@socher.org

**Samuel J. Gershman, Adler J. Perotte, Per B. Sederberg**
Department of Psychology
Princeton University
Princeton, NJ 08540
{sjgershm,aperotte,persed}@princeton.edu

**David M. Blei**
Department of Computer Science
Princeton University
Princeton, NJ 08540
blei@cs.princeton.edu

**Kenneth A. Norman**
Department of Psychology
Princeton University
Princeton, NJ 08540
knorman@princeton.edu

## Abstract

We develop a probabilistic model of human memory performance in free recall experiments. In these experiments, a subject first studies a list of words and then tries to recall them. To model these data, we draw on both previous psychological research and statistical topic models of text documents. We assume that memories are formed by assimilating the semantic meaning of studied words (represented as a distribution over topics) into a slowly changing latent context (represented in the same space). During recall, this context is reinstated and used as a cue for retrieving studied words. By conceptualizing memory retrieval as a dynamic latent variable model, we are able to use Bayesian inference to represent uncertainty and reason about the cognitive processes underlying memory. We present a particle filter algorithm for performing approximate posterior inference, and evaluate our model on the prediction of recalled words in experimental data. By specifying the model hierarchically, we are also able to capture inter-subject variability.

## 1   Introduction

Modern computational models of verbal memory assume that the recall of items is shaped by their semantic representations. The precise nature of this relationship is an open question. To address it, recent research has used information from diverse sources, such as behavioral data [14], brain imaging [13] and text corpora [8]. However, a principled framework for integrating these different types of information is lacking. To this end, we develop a model of human memory that encodes probabilistic dependencies between multiple information sources and the hidden variables that couple them. Our model lets us combine multiple sources of information and multiple related memory experiments.

Our model builds on the *Temporal Context Model* (TCM) of [10, 16]. TCM was developed to explain the temporal structure of human behavior in free recall experiments, where subjects are presented with lists of words (presented one at a time) and then asked to recall them in any order. TCM posits a slowly changing mental context vector whose evolution is driven by lexical input. At study, words are bound to context states through learning; during recall, context information is used as a cue to probe for stored words. TCM can account for numerous regularities in free recall data, most prominently the finding that subjects tend to consecutively recall items that were studied close in time to one another. (This effect is called the *temporal contiguity effect*.) TCM explains this effect by positing that recalling an item also triggers recall of the context state that was present when the

item was studied; subjects can use this retrieved context state to access items that were studied close in time to the just-recalled item. The fact that temporal contiguity effects in TCM are mediated indirectly (via item-context associations) rather than directly (via item-item associations) implies that temporal contiguity effects should persist when subjects are prevented from forming direct item-item associations; for evidence consistent with this prediction, see [9].

Importantly, temporal structure is not the only organizing principle in free recall data: Semantic relatedness between items also influences the probability of recalling them consecutively [11]. Moreover, subjects often recall semantically-related items that were not presented at study. (These are called extra-list intrusions; see [15].) To capture this semantic structure, we will draw on probabilistic topic models of text documents, specifically *latent Dirichlet allocation* (LDA) [3]. LDA is an unsupervised model of document collections that represents the meaning of documents in terms of a small number of "topics," each of which is a distribution over words. When fit to a corpus, the most probable words of these distributions tend to represent the semantic themes (like "sports" or "chemistry") that permeate the collection. LDA has been used successfully as a psychological model of semantic representation [7].

We model free recall data by combining the underlying assumptions of TCM with the latent semantic space provided by LDA. Specifically, we reinterpret TCM as a dynamic latent variable model where the mental context vector specifies a distribution over topics. In other words, the human memory component of our model represents the drifting mental context as a sequence of mixtures of topics, in the same way that LDA represents documents. With this representation, the dynamics of the mental context are determined by two factors: the posterior probability over topics given a studied or recalled word (semantic inference) and the retrieval of previous contexts (episodic retrieval). These dynamics let us capture both the episodic and semantic structure of human verbal memory.

The work described here goes beyond prior TCM modeling work in two ways: First, our approach allows us to infer the trajectory of the context vector over time, which (in turn) allows us to predict the item-by-item sequence of word recalls; by contrast, previous work (e.g., [10, 16]) has focused on fitting the summary statistics of the data. Second, we model inter-subject variability using a hierarchical model specification; this approach allows us to capture both common and idiosyncratic features of the behavioral data.

The rest of the paper is organized as follows. In Section 2 we describe LDA and in Section 3 we describe our model, which we refer to as LDA-TCM. In Section 4 we describe a particle filter for performing posterior inference in this model. In Section 5.1 we present simulation results showing how this model reproduces fundamental behavioral effects in free recall experiments. In Section 5.2 we present inference results for a dataset collected by Sederberg and Norman in which subjects performed free recall of words.

## 2 Latent Dirichlet allocation

Our model builds on probabilistic topic models, specifically latent Dirichlet allocation. Latent Dirichlet allocation (LDA) is a probabilistic model of document collections [3]. LDA posits a set of $K$ topics, each of which is a distribution over a fixed vocabulary, and documents are represented as mixtures over these topics. Thus, each word is assumed to be drawn from a mixture model with corpus-wide components (i.e., the topics) and document-specific mixture proportions. When fit to a collection of documents, the topic distributions often reflect the themes that permeate the document collection.

More formally, assume that there are $K$ topics $\beta_k$, each of which is a distribution over words. (We will call the $K \times W$ matrix $\beta$ the *word distribution matrix*.) For each document, LDA assumes the following generative process:

1. Choose topic proportions $\theta \sim \text{Dir}(\alpha)$.

2. For each of the $N$ words $w_n$:

   (a) Choose a topic assignment $z_n \sim \text{Mult}(\theta)$.

   (b) Choose a word $w_n \sim \text{Mult}(\beta_{z_n})$.

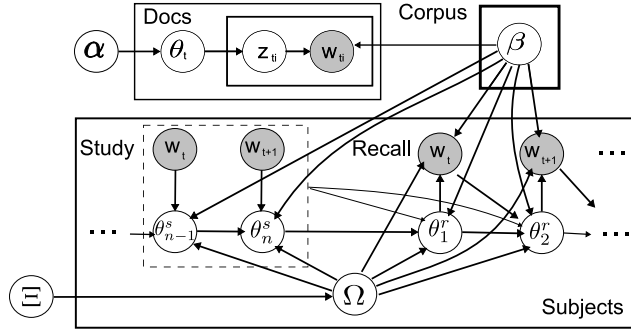

Figure 1: A graphical model of LDA-TCM.

Given a collection of documents, posterior inference in LDA essentially reverses this process to decompose the corpus according to its topics and find the corresponding distributions over words. Posterior inference is intractable, but many approximation algorithms have been developed [3, 7, 17].

In addition to capturing the semantic content of documents, recent psychological work has shown that several aspects of LDA make it attractive as a model of human semantic representation [7]. In our model of memory, the topic proportions $\chi$ play the role of a "mental context" that guides memory retrieval by parameterizing a distribution over words to recall.

## 3 Temporal context and memory

We now turn to a model of human memory that uses the latent representation of LDA to capture the semantic aspects of recall experiments. Our data consist of two types of observations: a corpus of documents from which we have obtained the word distribution matrix, [1] and behavioral data from free recall experiments, which are studied and recalled words from multiple subjects over multiple runs of the experiment. Our goal is to model the psychological process of recall in terms of a drifting mental context.

The human memory component of our model is based on the Temporal Context Model (TCM). There are two core principles of TCM: (1) Memory retrieval involves reinstating a representation of context that was active at the time of study; and (2) context change is driven by features of the studied stimuli [10, 16, 14]. We capture these principles by representing the mental context drift of each subject with a trajectory of latent variables $\chi_n$. Our use of the same variable name ($\chi$) and dimensionality for the context vector and for topics reflects our key assertion: Context and topics reside in the same meaning space.

The relationship between context and topics is specified in the generative process of the free recall data. The generative process encompasses both the study phase and the recall phase of the memory experiment. During study, the model specifies the distribution of the trajectory of internal mental contexts of the subject. (These variables are important in the next phase when recalling words episodically.) First, the initial mental context is drawn from a Gaussian:

$$\chi_{s,0} \sim \mathcal{N}(\mathbf{0}, \phi I) \tag{1}$$

where $s$ denotes the study phase and $I$ is a $K \times K$ identity matrix.[2] Then, for each studied word the mental context drifts according to

$$\chi_{s,n} \sim \mathcal{N}(\mathbf{h}_{s,n}, \phi I) \tag{2}$$

where

$$\mathbf{h}_{s,n} = \rho_1 \chi_{s,n-1} + (1 - \rho_1) \log(\mathbf{p}_{s,n}) \tag{3}$$

This equation identifies the two pulls on mental context drift when the subject is studying words: the previous context vector $\theta_{n-1}$ and $\tilde{\mathbf{p}}_{s,n} \propto \beta_{\cdot,w_{s,n}}$, the posterior probabilities of each topic given the current word and the topic distribution matrix. This second term captures the idea that mental context is updated with the meaning of the current word (see also [2] for a related treatment of topic dynamics in the context of text modeling). For example, if the studied word is "stocks" then the mental context might drift toward topics that also have words like "business", "financial", and "market" with high probability. (Note that this is where the topic model and memory model are coupled.) The parameter $\eta_1$ controls the rate of drift, while $\sigma$ controls its noisiness.

During recall, the model specifies a distribution over drifting contexts and recalled words. For each time $t$, the recalled word is assumed to be generated from a mixture of two components. Effectively, there are two "paths" to recalling a word: a semantic path and an episodic path.

The semantic path recalls words by "free associating" according to the LDA generative process: Using the current context as a distribution over topics, it draws a topic randomly and then draws a word from this topic (this is akin to thinking of a word that is *similar in meaning* to just-recalled words). Formally, the probability of recalling a word via the semantic path is expressed as the marginal probability of that word induced by the current context:

$$P_s(w) = \pi(\theta_{r,t}) \cdot \beta_{\cdot,w}, \tag{4}$$

where $\pi$ is a function that maps real-valued vectors onto the simplex (i.e., positive vectors that sum to one) and the index $r$ denotes the recall phase.

The episodic path recalls words by drawing them exclusively from the set of studied words. This path puts a high probability on words that were studied in a context that resembles the current context (this is akin to remembering words that you studied when you were thinking about things similar to what you are currently thinking about). Formally, the episodic distribution over words is expressed as a weighted sum of delta functions (each corresponding to a word distribution that puts all its mass on a single studied word), where the weight for a particular study word is determined by the similarity of the context at recall to the state of context when the word was studied:

$$P_e(w) = \frac{u_{t,w}}{\sum_i u_{t,i}}, \tag{5}$$

where

$$\mathbf{u}_t = \sum_{n=1}^{N} \delta_{s,w_{s,n}} / d(\pi(\theta_{r,t}), \pi(\theta_{s,n}))^{\epsilon}.$$

Here $d(\cdot, \cdot)$ is a similarity function between distributions (here we use the negative KL-divergence) and $\epsilon$ is a parameter controlling the curvature of the similarity function. We define $\{\delta_{s,w_{s,n}}\}_{n=1}^{N}$ to be delta functions defined at study words. Because people tend not to repeatedly recall words, we remove the corresponding delta function after a word is recalled.

Our model assumes that humans use some mixture of these two paths, determined by mixing proportion $\lambda$. Letting $w_{r,t} \sim \text{Mult}(\phi_t)$, we have

$$\phi_t(w) = \lambda P_s(w) + (1 - \lambda) P_e(w). \tag{6}$$

Intuitively, $\lambda$ in Equation 6 controls the balance between semantic influences and episodic influences. When $\lambda$ approaches 1, we obtain a "pure semantic" model wherein words are recalled essentially by free association (this is similar to the model used by [7] to model semantically-related intrusions in free recall). When $\lambda$ approaches 0, we obtain a "pure episodic" model wherein words are recalled exclusively from the study list. An intermediate value of $\lambda$ is essential to simultaneously explaining temporal contiguity and semantic effects in memory.

Finally, the context drifts according to

$$\theta_{r,t+1} \sim \mathcal{N}(\mathbf{h}_{r,t}, \sigma I), \tag{7}$$

where

$$\mathbf{h}_{r,t} = \eta_2 \theta_{r,t} + \eta_3 \log(\tilde{\mathbf{p}}_{r,t}) + \eta_4 \theta_{s,n(w_{r,t})}. \tag{8}$$

This is similar to how context drifts in the study phase, except that the context is additionally pushed by the context that was present when the recalled word was studied. This is obtained mathematically by defining $n(w_{r,t})$ to be a mapping from a recalled word to the index of the same word at study. For

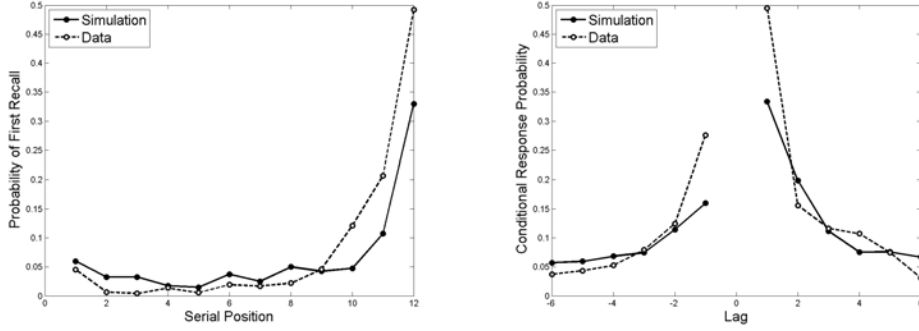

Figure 2: Simulated and empirical recall data. Data replotted from [9]. (*Left*) Probability of first recall curve. (*Right*) Conditional response probability curve.

example, if the recalled word is "cat" and cat was the sixth studied word then $n(w_{r,t}) = 6$. If there is a false recall, i.e., the subject recalls a word that was not studied, then $\theta_{s,n(w_{r,t})}$ is set to the zero vector.

This generative model is depicted graphically in Figure 1, where $\Omega = \{\eta_{1:4}, \sigma, \lambda, \epsilon\}$ represents the set of model parameters and $\Xi$ is the set of hyperparameters.

To model inter-subject variability, we extend our model hierarchically, defining group-level prior distributions from which subject-specific parameters are assumed to be drawn [6]. This approach allows for inter-subject variability and, at the same time, it allows us to gain statistical strength from the ensemble by coupling subjects in terms of higher-level hyperparameters. We choose our group prior over subject $i$'s parameters to factorize as follows:

$$P(\eta_{1:4}^i, \sigma^i, \lambda^i, \epsilon^i) = P(\eta_1^i)P(\eta_{2:4}^i)P(\sigma^i)P(\lambda^i)P(\epsilon^i). \tag{9}$$

In more detail, the factors take on the following functional forms: $\eta_1^i \sim \text{Beta}(c, d), \eta_{2:4}^i \sim \text{Dir}(\chi), \sigma^i \sim \text{Exp}(\nu), \lambda^i \sim \text{Beta}(a, b), \epsilon^i \sim \text{Gamma}(\alpha_1, \alpha_2)$. Except where mentioned otherwise, we used the following hyperparameter values: $a = b = c = d = 1, \chi = [1, 1, 1], \alpha_1 = 1, \alpha_2 = 1$. For some model variants (described in Section 5.2) we set the parameters to a fixed value rather than inferring them.

Here, we use the model to answer the following questions about behavior in free recall experiments: (1) Do both semantic and temporal factors influence recall, and if so what are their relative contributions; (2) What are the relevant dimensions of variation across subjects? In our model, semantic and temporal factors exert their influence via the context vector, while variation across subjects is expressed in the parameters drawn from the group prior. Thus, our goal in inference is to compute the posterior distribution over the context trajectory and subject-specific parameters, given a sequence of studied and recalled words. We can also use this posterior to make predictions about what words will be recalled by a subject at each point during the recall phase. By comparing the predictive performance of different model variants, we can examine what types of model assumptions (like the balance between semantic and temporal factors) best capture human behavior.

## 4 Inference

We now describe an approximate inference algorithm for computing the posterior distribution. Letting $\theta = \{\theta_{s,0:N}, \theta_{r,1:T}, \Omega\}$, the posterior is:

$$P(\theta|\mathbf{W}) = \frac{P(\mathbf{w}_{r,1:T}|\theta_{s,1:N}, \theta_{r,1:T}, \mathbf{w}_{s,1:N})P(\theta_{r,1:T}|\theta_{s,1:N})P(\theta_{s,1:N}|\mathbf{w}_{s,1:N}, \theta_{s,0})P(\theta_{s,0})P(\Omega)}{P(\mathbf{w}_{s,1:N}, \mathbf{w}_{r,1:T})}. \tag{10}$$

Because computing the posterior exactly is intractable (the denominator involves a high-dimensional integral that cannot be solved exactly), we approximate it with a set of $C$ samples using the particle filter algorithm [4], which can be summarized as follows. At time $t > 0$:

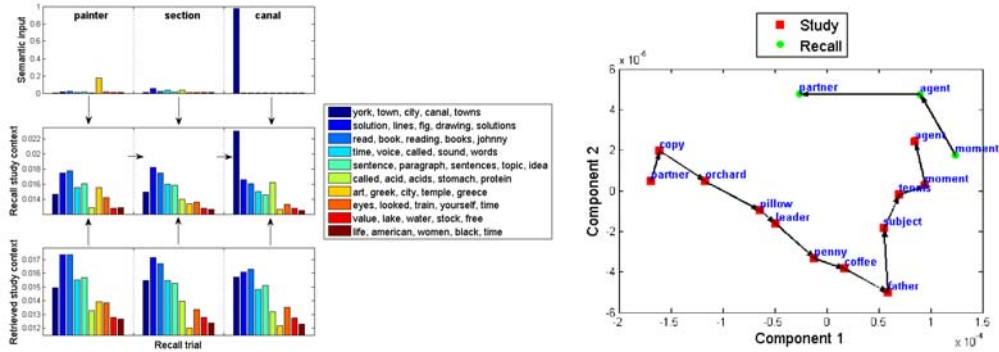

Figure 3: Factors contributing to context change during recall on a single list. (*Left*) Illustration of how three successively recalled words influence context. Each column corresponds to a specific recalled word (shown in the top row). The bars in each cell correspond to individual topics (specifically, these are the top ten inferred topics at recall; the center legend shows the top five words associated with each topic). Arrows schematically indicate the flow of influence between the components. The context vector at recall (*Middle Row*) is updated by the posterior over topics given the recalled word (*Top Row*) and also by retrieved study contexts (*Bottom Row*). (*Right*) Plot of the inferred context trajectory at study and recall for a different list, in a 2-dimensional projection of the context space obtained by principal components analysis.

1. Sample recall context $\theta_t^{(c)}$ using (7).

2. Compute weights $v_t^{(c)} \propto P\left(w_{r,t}|\theta_{r,t}^{(c)}\right)$ using (6).

3. Resample the particles according to their weights.

Using this sample-based approximation, the posterior is approximated as a sum of the delta functions placed at the samples:

$$P(\theta|\mathbf{W}) \approx \frac{1}{C}\sum_{c=1}^{C} \delta\left(\theta - \theta^{(c)}\right). \tag{11}$$

## 5  Results

We evaluate our model in two ways. First, we generate data from the generative model and record a number of common psychological measurements to assess to what extent the model reproduces qualitative patterns of recall behavior. Second, we perform posterior inference and evaluate the predictive performance of the model on a real dataset gathered by Sederberg and Norman.

### 5.1  Simulations

For the simulations, the following parameters were used: $\eta_1 = 0.2, \eta_2 = 0.55, \eta_3 = 0.05, \sigma = 0.00001, \lambda = 0.2, \epsilon = 1.7$. Note that these parameters have not been fit quantitatively to the data; here we are simply trying to reproduce qualitative patterns. These values have been chosen heuristically without a systematic search through the parameter space. The results are averaged over 400 random study lists of 12 words each. In Figure 3, we compare our simulation results to data collected by [9].

Figure 2 (left) shows the probability of first recall (PFR) curve, which plots the probability of each list position being the first recalled word. This curve illustrates how words in later positions are more likely to be recalled first, a consequence (in our model) of initializing the recall context with the last study context. Figure 2 (right) shows the lag conditional response probability (lag-CRP) curve, which plots the conditional probability of recalling a word given the last recalled word as a function of the lag (measured in terms of serial position) between the two. This curve demonstrates the temporal

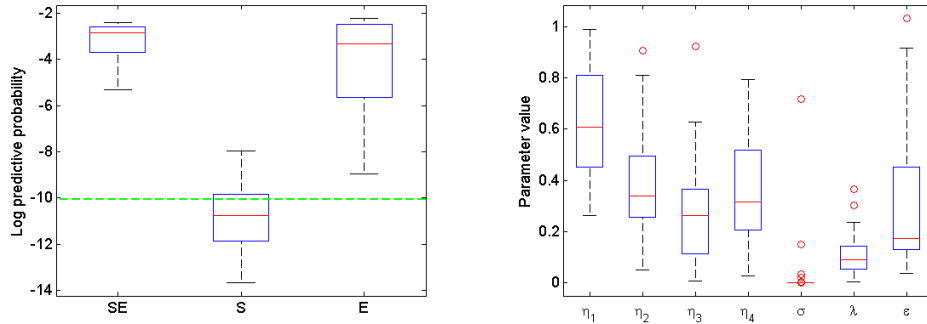

Figure 4: (*Left*) Box-plot of average predictive log-probability of recalled words under different models. S: pure semantic model; E: pure episodic model. Green line indicates chance. See text for more detailed descriptions of these models. (*Right*) Box-plot of inferred parameter values across subjects.

contiguity effect observed in human recall behavior: the increased probability of recalling words that were studied nearby in time to the last-recalled word. As in TCM, this effect is present in our model because items studied close in time to one another have similar context vectors; as such, cuing with contextual information from time $t$ will facilitate recall of other items studied in temporal proximity to time $t$.

## 5.2 Modeling psychological data

The psychological data modeled here are from a not-previously-published dataset collected by Sederberg and Norman. 30 participants studied 8 lists of words for a delayed free-recall task. Each list was composed of 15 common nouns, chosen at random and without replacement from one of 28 categories, such as Musical Instruments, Sports, or Four-footed Animals. After fitting LDA to the TASA corpus [5], we ran the particle filter with 1000 particles on the Sederberg and Norman dataset. Our main interest here is comparing our model (which we refer to as the semantic-episodic model) against various special hyperparameter settings that correspond to alternative psychological accounts of verbal memory. The models being compared include:

1. **Pure semantic**: defined by drawing words exclusively from the semantic path, with $\lambda = 1$. This type of model has been used by [7] to examine semantic similarity effects in free recall.

2. **Pure episodic**: defined by drawing words exclusively from the episodic path, with $\lambda = 0$.

3. **Semantic-episodic**: $a = b = 1$ (uniform beta prior on $\lambda$). This corresponds to a model in which words are drawn from a mixture of the episodic and semantic paths.

We also compare against a null (chance) model in which all words in the vocabulary have an equal probability of being recalled.

As a metric of model comparison, we calculate the model's predictive probability for the word recalled at time $t$ given words 1 to $t - 1$, for all $t$:

$$\sum_{t=1}^{T} -\log p(w_{r,t}|w_{r,1:t-1}, w_{s,1:N}). \tag{12}$$

This metric is proportional to the accumulative prediction error [19], a variant of cross-validation designed for time series models.

To assure ourselves that the particle filter we used does not suffer from weight degeneracy, we also calculated the *effective sample size*, as recommended by [4]: $ESS = \left(\sum_{c=1}^{C} \left(v^{(c)}\right)^2\right)^{-1}$. Conventionally, it is desirable that the effective sample size is at least half the number of particles. This desideratum was satisfied for all the models we explored.

Before we present the quantitative results, it is useful to examine some examples of inferred context change and how it interacts with word recall. Figure 3 shows the different factors at work in generating context change during recall on a single trial, illustrating how semantic inference and retrieved episodic memories combine to drive context change. The legend showing the top words in each topic illustrates how these topics appear to capture some of the semantic structure of the recalled words. On the right of Figure 3, we show another representation of context change (from a different trial), where the context trajectory is projected onto the first two principal components of the context vector. We can see from this figure how recall involves reinstatement of studied contexts: Recalling a word pulls the inferred context vector in the direction of the (inferred) contextual state associated with that word at study.

Figure 4 (left) shows the average predictive log-probability of recalled words for the models described above. Overall, the semantic-episodic model outperforms the pure episodic and pure semantic models in predictive accuracy (superiority over the closest competitor, the pure episodic model, was confirmed by a paired-sample t-test, with $p < 0.002$). To gain deeper insight into this pattern of results, consider the behavior of the different "pure" models with respect to extra-list intrusions vs. studied list items. The pure episodic model completely fails to predict extra-list intrusions, because it restricts recall to the study list (i.e., it assigns zero predictive probability to extra-list items). Conversely, the pure semantic model does a poor job of predicting recall of studied list items, because it does *not* scope recall to the study list. Thus, each of these models is hobbled by crucial (but complementary) shortcomings. The semantic-episodic model, by occupying an intermediate position between these two extremes, is able to capture both the semantic and temporal structure in free recall.

Our second goal in inference was to examine individual differences in parameter fits. Figure 4 (right) shows box-plots of the different parameters. In some cases there is substantial variability across subjects, such as for the similarity parameter $\epsilon$. Another pattern to notice is that the values of the episodic-semantic trade-off parameter $\lambda$ tend to cluster close to 0 (the episodic extreme of the spectrum), consistent with the fact that the pure episodic and semantic-episodic models are fairly comparable in predictive accuracy. Future work will assess the extent to which these across-subject differences in parameter fits reflect stable individual differences in memory functioning.

## 6 Discussion

We have presented here LDA-TCM, a probabilistic model of memory that integrates semantic and episodic influences on recall behavior. By formalizing this model as a probabilistic graphical model, we have provided a common language for developing and comparing more sophisticated variants. Our simulation and empirical results show that LDA-TCM captures key aspects of the experimental data and provides good accuracy at making item-by-item recall predictions. The source code for learning and inference and the experimental datasets are available at `www.cs.princeton.edu/~blei`.

There are a number of advantages to adopting a Bayesian approach to modeling free recall behavior. First, it is easy to integrate more sophisticated semantic models such as hierarchical Dirichlet processes [18]. Second, hierarchical model specification gives us the power to capture both common and idiosyncratic behavioral patterns across subjects, thereby opening a window onto individual differences in memory. Finally, this approach makes it possible to integrate other sources of data, such as brain imaging data. In keeping with the graphical model formalism, we plan to augment LDA-TCM with additional nodes representing variables measured with functional magnetic resonance imaging (fMRI). Existing studies have used fMRI data to decode semantic states in the brain [12] and predict recall behavior at the level of semantic categories [13]. Incorporating fMRI data into the model will have several benefits: The fMRI data will serve as an additional constraint on the inference process, thereby improving our ability to track subjects' mental states during encoding and recall; fMRI will give us a new way of validating the model – we will be able to measure the model's ability to predict both brain states and behavior; also, by examining the relationship between latent context states and fMRI data, we will gain insight into how mental context is instantiated in the brain.

**Acknowledgements**

RS acknowledges support from the Francis Robbins Upton Fellowship and the ERP Fellowship. This work was done while RS was at Princeton University. PBS acknowledges support from National Institutes of Health research grant MH080526.

## Footnotes

[1] For simplicity, we fix the word distribution matrix to one fit using the method of [3]. In future work, we will explore how the data from the free recall experiment could be used to constrain estimates of the word distribution matrix.

[2] More precisely, context vectors are log-transformed topic vectors (see [1, 2]). When generating words from the topics, we renormalize the context vector.

# References

[1] J. Aitchison. The statistical analysis of compositional data. *Journal of the Royal Statistical Society. Series B (Methodological)*, pages 139–177, 1982.

[2] D.M. Blei and J.D. Lafferty. Dynamic topic models. In *Proceedings of the 23rd international conference on Machine learning*, pages 113–120. ACM New York, NY, USA, 2006.

[3] D.M. Blei, A.Y. Ng, and M.I. Jordan. Latent dirichlet allocation. *Journal of Machine Learning Research*, 3:993–1022, 2003.

[4] A. Doucet and N. De Freitas. *Sequential Monte Carlo Methods in Practice*. Springer, 2001.

[5] ST Dumais and TK Landauer. A solution to Platos problem: The latent semantic analysis theory of acquisition, induction and representation of knowledge. *Psychological Review*, 104:211–240, 1997.

[6] A. Gelman and J. Hill. *Data analysis using regression and multilevel/hierarchical models*. Cambridge University Press, 2007.

[7] T.L. Griffiths, M. Steyvers, and J.B. Tenenbaum. Topics in semantic representation. *Psychological Review*, 114(2):211–244, 2007.

[8] M.W. Howard, B. Jing, K.M. Addis, and M.J. Kahana. Semantic structure and episodic memory. *Handbook of Latent Semantic Analysis*, pages 121–142, 2007.

[9] M.W. Howard and M.J. Kahana. Contextual variability and serial position effects in free recall. *Journal of Experimental Psychology: Learning, Memory, and Cognition*, 25(4):923, 1999.

[10] M.W. Howard and M.J. Kahana. A distributed representation of temporal context. *Journal of Mathematical Psychology*, 46:269–299, 2002.

[11] M.W. Howard and M.J. Kahana. When does semantic similarity help episodic retrieval? *Journal of Memory and Language*, 46(1):85–98, 2002.

[12] T.M. Mitchell, S.V. Shinkareva, A. Carlson, K. Chang, V.L. Malave, R.A. Mason, and M.A. Just. Predicting human brain activity associated with the meanings of nouns. *Science*, 320(5880):1191–1195, 2008.

[13] S.M. Polyn, V.S. Natu, J.D. Cohen, and K.A. Norman. Category-specific cortical activity precedes retrieval during memory search. *Science*, 310(5756):1963–1966, 2005.

[14] S.M. Polyn, K.A. Norman, and M.J. Kahana. A context maintenance and retrieval model of organizational processes in free recall. *Psychological Review*, 116(1):129, 2009.

[15] H.L. Roediger and K.B. McDermott. Creating false memories: Remembering words not presented in lists. *Journal of Experimental Psychology Learning Memory and Cognition*, 21:803–803, 1995.

[16] P.B. Sederberg, M.W. Howard, and M.J. Kahana. A context-based theory of recency and contiguity in free recall. *Psychological Review*, 115(4):893–912, 2008.

[17] Y. Teh, D. Newman, and M. Welling. A collapsed variational Bayesian inference algorithm for latent Dirichlet allocation. In *Neural Information Processing Systems*, 2006.

[18] Y.W. Teh, M.I. Jordan, M.J. Beal, and D.M. Blei. Hierarchical dirichlet processes. *Journal of the American Statistical Association*, 101(476):1566–1581, 2006.

[19] E.J. Wagenmakers, P. Grünwald, and M. Steyvers. Accumulative prediction error and the selection of time series models. *Journal of Mathematical Psychology*, 50(2):149–166, 2006.

